# Supervised Exponential Family Principal Component Analysis via Convex Optimization

**Yuhong Guo**
Computer Sciences Laboratory
Australian National University
yuhongguo.cs@gmail.com

## Abstract

Recently, supervised dimensionality reduction has been gaining attention, owing to the realization that data labels are often available and indicate important underlying structure in the data. In this paper, we present a novel convex supervised dimensionality reduction approach based on exponential family PCA, which is able to avoid the local optima of typical EM learning. Moreover, by introducing a sample-based approximation to exponential family models, it overcomes the limitation of the prevailing Gaussian assumptions of standard PCA, and produces a kernelized formulation for nonlinear supervised dimensionality reduction. A training algorithm is then devised based on a subgradient bundle method, whose scalability can be gained using a coordinate descent procedure. The advantage of our global optimization approach is demonstrated by empirical results over both synthetic and real data.

## 1 Introduction

Principal component analysis (PCA) has been extensively used for data analysis and processing. It provides a closed-form solution for linear unsupervised dimensionality reduction through singular value decomposition (SVD) on the data matrix [8]. Probabilistic interpretations of PCA have also been provided in [9, 16], which formulate PCA using a latent variable model with Gaussian distributions. To generalize PCA to better suit non-Gaussian data, many extensions to PCA have been proposed that relax the assumption of a Gaussian data distribution. Exponential family PCA is the most prominent example, where the underlying dimensionality reduction principle of PCA is extended to the general exponential family [4, 7, 13]. Previous work has shown that improved quality of dimensionality reduction can be obtained by using exponential family models appropriate for the data at hand [4, 13]. Given data from a non-Gaussian distribution these techniques are better able than PCA to capture the intrinsic low dimensional structure. However, most existing non-Gaussian dimensionality reduction methods rely on iterative local optimization procedures and thus suffer from local optima, with the sole exception of [7] which shows a general convex form can be obtained for dimensionality reduction with exponential family models.

Recently, supervised dimensionality reduction has begun to receive increased attention. As the goal of dimensionality reduction is to identify the intrinsic structure of a data set in a low dimensional space, there are many reasons why supervised dimensionality reduction is a meaningful topic to study. First, data labels are almost always assigned based on some important intrinsic property of the data. Such information should be helpful to suppress noise and capture the most useful aspects of a compact representation of the data. Moreover, there are many high dimensional data sets with label information available, e.g., face and digit images, and it is unwise to ignore them. A few supervised dimensionality reduction methods based on exponential family models have been proposed in the literature. For example, a supervised probabilistic PCA (SPPCA) model was proposed in [19]. SPPCA extends probabilistic PCA by assuming that both features and labels have Gaussian

distributions and are generated independently from the latent low dimensional space through linear transformations. The model is learned by maximizing the marginal likelihood of the observed data using an alternating EM procedure. A more general supervised dimensionality reduction approach with generalized linear models (SDR_GLM) was proposed in [12]. SDR_GLM views both features and labels as exponential family random variables and optimizes a weighted linear combination of their conditional likelihood given latent low dimensional variables using an alternating EM-style procedure with closed-form update rules. SDR_GLM is able to deal with different data types by using different exponential family models. Similar to SDR_GLM, the linear supervised dimensionality reduction method proposed in [14] also takes advantage of exponential family models to deal with different data types. However, it optimizes the conditional likelihood of labels given observed features within a mixture model framework using an EM-style optimization procedure. Beyond the PCA framework, many other supervised dimensionality reduction methods have been proposed in the literature. Linear (fisher) discriminant analysis (LDA) is a popular alternative [5], which maximizes between-class variance and minimizes within-class variance. Moreover, a kernelized fisher discriminant analysis (KDA) has been studied in [10]. Another notable nonlinear supervised dimensionality reduction approach is the colored maximum variance unfolding (MVU) approach proposed in [15], which maximizes the variance aligning with the side information (e.g., label information), while preserving the local distance structures from the data. However, colored MVU has only been evaluated on training data.

In this paper, we propose a novel supervised exponential family PCA model (SEPCA). In the SEPCA model, observed data $\mathbf{x}$ and its label $y$ are assumed to be generated from the latent variables $\mathbf{z}$ via conditional exponential family models; dimensionality reduction is conducted by optimizing the conditional likelihood of the observations $(\mathbf{x}, y)$. By exploiting convex duality of the sub-problems and eigenvector properties, a solvable convex formulation of the problem can be derived that preserves solution equivalence to the original. This convex formulation allows efficient global optimization algorithms to be devised. Moreover, by introducing a sample-based approximation to exponential family models, SEPCA does not suffer from the limitations of implicit Gaussian assumptions and is able to be conveniently kernelized to achieve nonlinearity. A training algorithm is then devised based on a subgradient bundle method, whose scalability can be gained through a coordinate descent procedure. Finally, we present a simple formulation to project new testing data into the embedded space. This projection can be used for other supervised dimensionality reduction approach as well. Our experimental results over both synthetic and real data suggest that a more global, principled probabilistic approach, SEPCA, is better able to capture subtle structure in the data, particularly when good label information is present.

The remainder of this paper is organized as follows. First, in Section 2 we present the proposed supervised exponential family PCA model and formulate a convex nondifferentiable optimization problem. Then, an efficient global optimization algorithm is presented in Section 3. In Section 4, we present a simple projection method for new testing points. We then present the experimental results in Section 5. Finally, in Section 6 we conclude the paper.

## 2   Supervised Exponential Family PCA

We assume we are given a $t \times n$ data matrix, $X$, consisting of $t$ observations of $n$-dimensional feature vectors, $X_{i:}$, and a $t \times k$ indicator matrix, $Y$, with each row to indicate the class label for each observation $X_{i:}$; thus $\sum_{j=1}^{k} Y_{ij} = 1$. For simplicity, we assume features in $X$ are centered; that is, their empirical means are zeros. We aim to recover a $d$-dimensional re-representation, a $t \times d$ matrix $Z$, of the data $(d < n)$. This is typically viewed as discovering a latent low dimensional manifold in the high dimensional feature space. Since the label information $Y$ is exploited in the discovery process, this is called supervised dimensionality reduction. For recovering $Z$, a key restriction that one would like to enforce is that the features used for coding, $Z_{:j}$, should be linearly independent; that is, one would like to enforce the constraint $Z^{\top} Z = I$, which ensures that the codes are expressed by orthogonal features in the low dimensional representation.

Given the above setup, in this paper, we are attempting to address the problem of supervised dimensionality reduction using a probabilistic latent variable model. Our intuition is that the important intrinsic structure (underlying feature representation) of the data should be able to accurately generate/predict the original data features and labels.

In this section, we formulate the low-dimensional principal component discovering problem as a conditional likelihood maximization problem based on exponential family model representations, which can be reformulated into an equivalent nondifferentiable convex optimization problem. We then exploit a sample-based approximation to unify exponential family models for different data types.

## 2.1 Convex Formulation of Supervised Exponential Family PCA

As with the generalized exponential family PCA [4], we attempt to find low-dimensional representation by maximizing the conditional likelihood of the observation matrix $X$ and $Y$ given the latent matrix $Z$, $\log P(X, Y|Z) = \log P(X|Z) + \log P(Y|Z)$. Using the general exponential family representation, a regularized version of this maximization problem can be formulated as

$$
\max_{Z: Z^\top Z = I} \max_{W, \Omega, \mathbf{b}} \log P(X|Z, W) - \frac{\beta}{2} \text{tr}\left(WW^\top\right) + \log P(Y|Z, \Omega, \mathbf{b}) - \frac{\beta}{2}\left(\text{tr}\left(\Omega\Omega^\top\right) + \mathbf{b}^\top \mathbf{b}\right)
$$

$$
= \max_{Z: Z^\top Z = I} \max_{W, \Omega, \mathbf{b}} \text{tr}\left(ZWX^\top\right) - \sum_i \left(A(Z_{i:}, W) - \log P_0(X_{i:})\right) - \frac{\beta}{2}\text{tr}\left(WW^\top\right) \tag{1}
$$

$$
+ \text{tr}\left(Z\Omega Y^\top\right) + \mathbf{1}^\top Y\mathbf{b} - \sum_i A(Z_{i:}, \Omega, \mathbf{b}) - \frac{\beta}{2}\left(\text{tr}\left(\Omega\Omega^\top\right) + \mathbf{b}^\top \mathbf{b}\right)
$$

where $W$ is a $d \times n$ parameter matrix for conditional model $P(X|Z)$; $\Omega$ is a $d \times k$ parameter matrix for conditional model $P(Y|Z)$ and $\mathbf{b}$ is a $k \times 1$ bias vector; $\mathbf{1}$ denotes the vector of all 1s; $A(Z_{i:}, W)$ and $A(Z_{i:}, \Omega, \mathbf{b})$ are the log normalization functions to ensure valid probability distributions:

$$
A(Z_{i:}, W) = \log \int \exp\left(Z_{i:}W\mathbf{x}\right) P_0(\mathbf{x})\, d\mathbf{x} \,. \tag{2}
$$

$$
A(Z_{i:}, \Omega, \mathbf{b}) = \log \sum_{\ell=1}^{k} \exp\left(Z_{i:}\Omega\mathbf{1}_\ell + \mathbf{1}_\ell^\top \mathbf{b}\right) \tag{3}
$$

where $\mathbf{1}_\ell$ denotes a zero vector with a single 1 in the $\ell$th entry.

Note that the class variable $y$ is discrete, thus maximizing $\log P(Y|Z, \Omega, \mathbf{b})$ is a discriminative classification training. In fact, the second part of the objective function in (1) is simply a multi-class logistic regression. That is why we have incorporated an additional bias term $\mathbf{b}$ into the model.

**Theorem 1** *The optimization problem (1) is equivalent to*

$$
\min_{U^x, U^y} \max_{M: I \succeq M \succeq 0,\, tr(M) = d} \sum_i \left(A^*(U_{i:}^x) + \log P_0(X_{i:})\right) + \frac{1}{2\beta}tr\left((X - U^x)(X - U^x)^\top M\right)
$$

$$
+ \sum_i A^*(U_{i:}^y) + \frac{1}{2\beta}tr\left((Y - U^y)(Y - U^y)^\top (M + E)\right) \tag{4}
$$

*where $E$ is a $t \times t$ matrix with all 1s; $U^x$ is a $t \times n$ matrix; $U^y$ is a $t \times k$ matrix; $A^*(U_{i:}^x)$ and $A^*(U_{i:}^y)$ are the Fenchel conjugates of $A(Z_{i:}, W)$ and $A(Z_{i:}, \Omega, \mathbf{b})$ respectively; $M = ZZ^\top$ and $Z$ can be recovered by taking the top $d$ eigenvectors of $M$; and the model parameters $W, \Omega, \mathbf{b}$ can be recovered by*

$$
W = \frac{1}{\beta}Z^\top(X - U^x), \quad \Omega = \frac{1}{\beta}Z^\top(Y - U^y), \quad \mathbf{b} = \frac{1}{\beta}(Y - U^y)^\top \mathbf{1}
$$

*Proof:* The proof is simple and based on standard results. Due to space limitation, we only provide a summarization of the key steps here. There are three steps. The first step is to derive the Fenchel conjugate dual for each log partition function, $A(Z, .)$, following [18, Section 3.3.3]; which can be used to yield

$$
\max_{Z: Z^\top Z = I} \min_{U^x, U^y} \sum_i \left(A^*(U_{i:}^x) + \log P_0(X_{i:})\right) + \frac{1}{2\beta}\text{tr}\left((X - U^x)(X - U^x)^\top ZZ^\top\right)
$$

$$
+ \sum_i A^*(U_{i:}^y) + \frac{1}{2\beta}\text{tr}\left((Y - U^y)(Y - U^y)^\top (ZZ^\top + E)\right) \tag{5}
$$

that is equivalent to the original problem (1). The second step is based on exploiting the strong min-max property [2] and the relationships between different constraint sets

$$\{M : M = ZZ^\top \text{ for some } Z \text{ such that } Z^\top Z = I\} \subseteq \{M : I \succeq M \succeq 0, \text{tr}(M) = d\},$$

which allows one to further show the optimization (4) is an upper bound relaxation of (5). The final equivalence proof is based on the result of [11], which suggests the substitution of $ZZ^\top$ with matrix $M$ does not produce relaxation gap. ∎

Note that (4) is a min-max optimization problem. Moreover, for each fixed $M$, the outer minimization problem is obviously convex, since the Fenchel conjugates, $A^*(U_{i:}^x)$ and $A^*(U_{i:}^y)$, are convex functions of $U^x$ and $U^y$ respectively [2]; that is, the objective function for the outer minimization is a pointwise supremum over an infinite set of convex functions. Thus the overall min-max optimization is convex [3], but apparently not necessarily differentiable. We will address the nondifferentiable training issue in Section 3.

## 2.2 Sample-based Approximation

In the previous section, we have formulated our supervised exponential family PCA as a convex optimization problem (4). However, before attempting to devise a training algorithm to solve it, we have to provide some concrete forms for the Fenchel conjugate functions $A^*(U_{i:}^x)$ and $A^*(U_{i:}^y)$. For different exponential family models, the Fenchel conjugate functions $A^*$ are different; see [18, Table 2]. For example, since the $y$ variable in our model is a discrete class variable, it takes a multinomial distribution. Thus the Fenchel conjugate function $A^*(U_{i:}^y)$ is given by

$$A^*(U_{i:}^y) = A^*(\Theta_{i:}^y) = \text{tr}\left(\Theta_{i:}^y \log \Theta_{i:}^{y\top}\right), \text{ where } \Theta^y \geq 0, \ \Theta^y \mathbf{1} = \mathbf{1} \tag{6}$$

The specific exponential family model is determined by the data type and distribution. PCA and SPPCA use Gaussian models, thus their performances might be degraded when the data distribution is non-Gaussian. However, it is tedious and sometimes hard to choose the most appropriate exponential family model to use for each specific application problem. Moreover, the log normalization function $A$ and its Fenchel conjugate $A^*$ might not be easily computable. For these reasons, we propose to use a sample-based approximation to the integral (2) and achieve an empirical approximation to the true underlying exponential family model as follows. If one replaces the integral definition (2) with an empirical definition, $A(Z_{i:}, W) = \log \sum_j \exp\left(Z_{i:} W X_{j:}^\top\right)/t$, then the conjugate function can be given by

$$A^*(U_{i:}^x) = A^*(\Theta_{i:}^x) = \text{tr}\left(\Theta_{i:}^x \log \Theta_{i:}^{x\top}\right) - \log(1/t), \text{ where } \Theta^x \geq 0, \ \Theta^x \mathbf{1} = \mathbf{1} \tag{7}$$

With this sample-based approximation, problem (4) can be expressed as

$$\min_{\Theta^x, \Theta^y} \max_{M:I \succeq M \succeq 0, \ \text{tr}(M)=d} \text{tr}\left(\Theta^x \log \Theta^x\right) + \frac{1}{2\beta} \text{tr}\left((I-\Theta^x)K(I-\Theta^x)^\top M\right) \tag{8}$$

$$+ \text{tr}\left(\Theta^y \log \Theta^y\right) + \frac{1}{2\beta} \text{tr}\left((Y-\Theta^y)(Y-\Theta^y)^\top (M+E)\right)$$

$$\text{subject to} \quad \Theta^x \geq 0, \ \Theta^x \mathbf{1} = \mathbf{1}; \ \Theta^y \geq 0, \ \Theta^y \mathbf{1} = \mathbf{1} \tag{9}$$

One benefit of working with this sample-based approximation is that it is automatically kernelized, $K = XX^\top$, to enable non-linearity to be conveniently introduced.

## 3 Efficient Global Optimization

The optimization (8) we derived in the previous section is a convex-concave min-max optimization problem. The inner maximization of (8) is a well known problem with a closed-form solution [11]: $M^* = Z^* Z^{*\top}$ and $Z^* = Q_{max}^d\left((I-\Theta^x)K(I-\Theta^x)^\top + (Y-\Theta^y)(Y-\Theta^y)^\top\right)$, where $Q_{max}^d(D)$ denotes the matrix formed by the top $d$ eigenvectors of $D$. However, the overall outer minimization problem is nondifferentiable with respect to $\Theta^x$ and $\Theta^y$. Thus the standard first-order or second-order optimization techniques that rely on the standard gradients can not be applied here. In this section, we deploy a bundle method to solve this nondifferentiable min-max optimization.

## 3.1 Bundle Method for Min-Max Optimization

The bundle method is an efficient subgradient method for nondifferentiable convex optimization; it relies on the computation of subgradient terms of the objective function. A vector $\mathbf{g}$ is a **subgradient** of function $f$ at point $\mathbf{x}$, if $f(\mathbf{y}) \geq f(\mathbf{x}) + \mathbf{g}^\top(\mathbf{y} - \mathbf{x}), \forall \mathbf{y}$. To adapt standard bundle methods to our specific min-max problem, we need to first address the critical issue of subgradient computation.

**Proposition 1** *Consider a joint function $h(\mathbf{x}, \mathbf{y})$ defined over $\mathbf{x} \in \mathcal{X}$ and $\mathbf{y} \in \mathcal{Y}$, satisfying: (1) $h(\cdot, \mathbf{y})$ is convex for all $\mathbf{y} \in \mathcal{Y}$; (2) $h(\mathbf{x}, \cdot)$ is concave for all $\mathbf{x} \in \mathcal{X}$. Let $f(\mathbf{x}) = \max_{\mathbf{y}} h(\mathbf{x}, \mathbf{y})$, and $q(\mathbf{x}_0) = \arg\max_{\mathbf{y}} h(\mathbf{x}_0, \mathbf{y})$. Assume that $\mathbf{g}$ is a gradient of $h(\cdot, q(\mathbf{x}_0))$ at $\mathbf{x} = \mathbf{x}_0$, then $\mathbf{g}$ is a subgradient of $f(\mathbf{x})$ at $\mathbf{x} = \mathbf{x}_0$.*

*Proof:*

$$
\begin{aligned}
f(\mathbf{x}) &= \max_{\mathbf{y}} h(\mathbf{x}, \mathbf{y}) \geq h(\mathbf{x}, q(\mathbf{x}_0)) \\
&\geq h(\mathbf{x}_0, q(\mathbf{x}_0)) + \mathbf{g}^\top(\mathbf{x} - \mathbf{x}_0) \quad \text{(since } h(\cdot, \mathbf{y}) \text{ is convex for all } \mathbf{y} \in \mathcal{Y}) \\
&= f(\mathbf{x}_0) + \mathbf{g}^\top(\mathbf{x} - \mathbf{x}_0) \quad \text{(by the definitions of } f(\mathbf{x}) \text{ and } q(\mathbf{x}_0))
\end{aligned}
$$

Thus $\mathbf{g}$ is a subgradient of $f(\mathbf{x})$ at $\mathbf{x} = \mathbf{x}_0$ according to the definition of subgradient. ∎

According to Proposition 1, the subgradients of our outer minimization objective function $f$ in (8) over $\Theta^x$ and $\Theta^y$ can be given by

$$
\partial_{\Theta^x} f \ni \left(\log \Theta^x + 1 - \frac{1}{\beta} M^*(I - \Theta^x)K\right), \quad \partial_{\Theta^y} f \ni \left(\log \Theta^y + 1 - \frac{1}{\beta} M^*(Y - \Theta^y)\right) \tag{10}
$$

where $M^*$ is the optimal inner maximization solution at the current point $[\Theta^x, \Theta^y]$.

Algorithm 1 illustrates the bundle method we developed to solve the infinite min-max optimization (8), where the linear constraints (9) over $\Theta^x$ and $\Theta^y$ can be conveniently incorporated into the quadratic bound optimization. One important issue in this algorithm is how to manage the size of the linear lower bound constraints formed from the active set $\mathbf{B}$ (defined in Algorithm 1), as it incrementally increases with new points being explored. To solve this problem, we noticed the Lagrangian dual parameters $\boldsymbol{\alpha}$ for the lower bound constraints obtained by the quadratic optimization in step 1 is a sparse vector, indicating that many lower bound constraints can be turned off. Moreover, any constraint that is turned off will mostly stay off in the later steps. Therefore, for the bundle method we developed, whenever the size of $\mathbf{B}$ is larger than a given constant $b$, we will keep the active points of $\mathbf{B}$ that correspond to the first $b$ largest $\boldsymbol{\alpha}$ values, and drop the remaining ones.

## 3.2 Coordinate Descent Procedure

An important factor affecting the running efficiency is the size of the problem. The convex optimization (8) works in the dual parameter space, where the size of the parameters $\Theta = \{\Theta^x, \Theta^y\}$, $t \times (t + k)$, depends only on the number of training samples, $t$, not on the feature size, $n$. For high dimensional small data sets ($n \gg t$), our dual optimization is certainly a good option. However, with the increase of $t$, our problem size will increase in an order of $O(t^2)$. It might soon become too large to handle for the quadratic optimization step of the bundle method.

On the other hand, the optimization problem (8) possesses a nice semi-decomposable structure: one equality constraint in (9) involves only one row of the $\Theta$; that is, the $\Theta$ can be separated into rows without affecting the equality constraints. Based on this observation, we develop a coordinate descent procedure to obtain scalability of the bundle method over large data sets. Specifically, we put an outer loop above the bundle method. Within each of this outer loop iteration, we randomly separate the $\Theta$ parameters into $m$ groups, with each group containing a subset rows of $\Theta$; and we then use bundle method to sequentially optimize each subproblem defined on one group of $\Theta$ parameters while keeping the remaining rows of $\Theta$ fixed. Although coordinate descent with a nondifferentiable convex objective is not guaranteed to converge to a minimum in general [17], we have found that this procedure performs quite well in practice, as shown in the experimental results.

# 4 Projection for Testing Data

One important issue for supervised dimensionality reduction is to map new testing data into the dimensionality-reduced principal dimensions. We deploy a simple procedure for this purpose. After

**Algorithm 1** Bundle Method for Min-Max Optimization in (8)

---

**Input:** $\bar{\delta} > 0, m \in (0,1), b \in I\!N, \mu \in I\!R$

**Initial:** Find an initial point $\theta^*$ satisfying the linear constraints in (9); compute $f(\theta^*)$.
Let $\ell = 1, \theta^\ell = \theta^*$, compute $\mathbf{g}^\ell \in \partial_{\theta^\ell} f$ by (10); $e^\ell = f(\theta^*) - f(\theta^\ell) - \mathbf{g}^{\ell\top}(\theta^* - \theta^\ell)$.
Let $\mathbf{B} = \{(e^\ell, \mathbf{g}^\ell)\}, \hat{\varepsilon} = \text{Inf}, \hat{\mathbf{g}} = \mathbf{0}; \ell = \ell + 1$.

**repeat**

1. Solve quadratic minimization for solution $\hat{\theta}$, and Lagrangian dual parameters $\boldsymbol{\alpha}$ w.r.t. the lower bound linear constraints in $\mathbf{B}$ [1]:

$$\hat{\theta} = \arg\min_\theta \psi_\ell(\theta) + \frac{\mu}{2}\|\theta - \theta^*\|^2, \text{ subject to the linear constraints in (9)}$$

where $\psi_\ell(\theta) = f(\theta^*) + \max\left\{ -\hat{\varepsilon} + \hat{\mathbf{g}}^\top(\theta - \theta^*), \max_{(e^i, \mathbf{g}^i) \in \mathbf{B}}\{-e^i + \hat{\mathbf{g}}^{i\top}(\theta - \theta^*)\}\right\}$

2. Define $\delta_\ell = f(\theta^*) - [\psi_\ell(\hat{\theta}) + \frac{\mu}{2}\|\hat{\theta} - \theta^*\|^2] \geq 0$. If $\delta_\ell < \hat{\delta}$, return.
3. Conduct line search to minimize $f(\theta^\ell)$ with $\theta^\ell = \gamma\theta^* + (1 - \gamma)\hat{\theta}$, for $0 < \gamma < 1$.
4. Compute $\mathbf{g}^\ell \in \partial_{\theta^\ell} f$ by (10); $e^\ell = f(\theta^*) - f(\theta^\ell) - \mathbf{g}^{\ell\top}(\theta^* - \theta^\ell)$; update $\mathbf{B} = \mathbf{B} \cup \{(e^\ell, \mathbf{g}^\ell)\}$.
5. **If** $f(\theta^*) - f(\theta^\ell) \geq m\delta_\ell$, **then** take a serious step:
   (1) update: $e^i = e^i + f(\theta^\ell) - f(\theta^*) + \mathbf{g}^{i\top}(\theta^* - \theta^\ell)$;
   (2) update the aggregation: $\hat{\mathbf{g}} = \sum_i \alpha_i \mathbf{g}^i, \hat{\varepsilon} = \sum_i \alpha_i e^i$;
   (3) update the stored solution: $\theta^* = \theta^\ell, f(\theta^*) = f(\theta^\ell)$.
6. If $|\mathbf{B}| > b$, reduce $\mathbf{B}$ set according to $\boldsymbol{\alpha}$.
7. $\ell = \ell + 1$.

**until** maximum iteration number is reached

---

training, we obtain a low-dimensional representation $Z$ for $X$, where $Z$ can be viewed as a linear projection of $X$ in some transformed space $\psi(X)$ through a parameter matrix U; such that $Z = \psi(X)U = \psi(X)\psi(X)^\top K^+ \psi(X)U$, where $K^+$ denotes the pseudo inverse of $K = \psi(X)\psi(X)^\top$. Then a new testing sample $\mathbf{x}^*$ can be projected by

$$\mathbf{z}^* = \psi(\mathbf{x}^*)\psi(X)^\top K^+ \psi(X)U = k(\mathbf{x}^*, X)K^+ Z \tag{11}$$

## 5 Experimental Results

In order to evaluate the performance of the proposed supervised exponential family PCA (SEPCA) approach, we conducted experiments over both synthetic and real data, and compared to supervised dimensionality reduction with generalized linear models (SDR_GLM), supervised probabilistic PCA (SPPCA), linear discriminant analysis (LDA), and colored maximum variance unfolding (MVU). The projection procedure (11) is used for colored MVU as well. In all the experiments, we used $\mu = 1$ for Algorithm 1, and used $\alpha = 0.0001$ for SDR_GLM as suggested in [12].

### 5.1 Experiments on Synthetic Data

Two synthetic experiments were conducted to compare the five approaches under controlled conditions. The first synthetic data set is formed by first generating four Gaussian clusters in a two-dimensional space, with each corresponding to one class, and then adding the third dimension to each point by uniformly sampling from a fixed interval. This experiment attempts to compare the performance of the five approaches in the situation where the data distribution does not satisfy the Gaussian assumption. Figure 1 shows the projection results for each approach in a two dimensional space for 120 testing points after being trained on a set with 80 points. In this case, SEPCA and LDA outperform all the other three approaches.

The second synthetic experiment is designed to test the capability of performing nonlinear dimensionality reduction. The synthetic data is formed by first generating two circles in a two dimensional space (one circle is located inside the other one), with each circle corresponding to one class, and then the third dimension sampled uniformly from a fixed interval. As SDR_GLM does not provide a nonlinear form, we conducted the experiment with only the remaining four approaches. For LDA, we used its kernel variant, KDA. A Gaussian kernel with $\sigma = 1$ was used for SEPCA, SPPCA and KDA. Figure 2 shows the projection results for each approach in a two dimensional space for 120

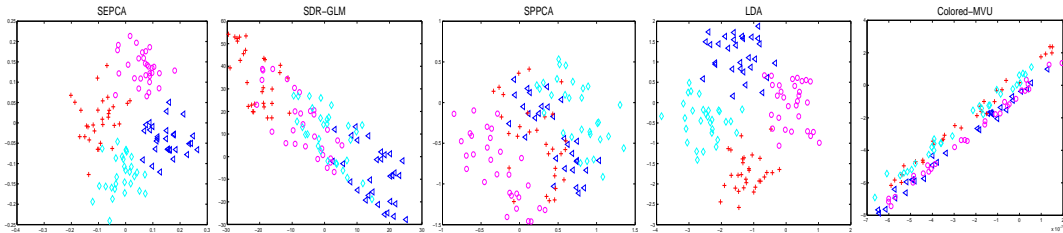

Figure 1: Projection results on test data for synthetic experiment 1. Each color indicates one class.

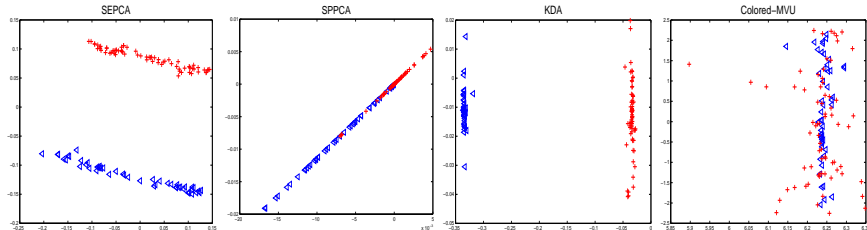

Figure 2: Projection results on test data for synthetic experiment 2. Each color indicates one class.

testing points after being trained on a set with 95 points. Again, SEPCA and KDA achieve good class separations and outperform the other two approaches.

## 5.2 Experiments on Real Data

To better characterize the performance of dimensionality reduction in a supervised manner, we conducted some experiments on a few high dimensional multi-class real world data sets. The left side of Table 1 provides the information about these data sets. Our experiments were conducted in the following way. We randomly selected 3∼5 examples from each class to form the training set and used the remaining examples as the test set. For each approach, we first learned the dimensionality reduction model on the training set. Moreover, we also trained a logistic regression classifier using the projected training set in the reduced low dimensional space. (Note, for SEPCA, a classifier was trained simultaneously during the process of dimensionality reduction optimization.) Then the test data were projected into the low dimensional space according to each dimensionality reduction model. Finally, the projected test set for each approach were classified using each corresponding logistic regression classifier. The right side of Table 1 shows the classification accuracies on the test set for each approach. To better understand the quality of the classification using projected data, we also included the standard classification results, indicated as 'FULL', using the original high dimensional data. (Note, we are not able to obtain any result for SDR_GLM on the newsgroup data as it is inefficient for very high dimensional data.) The results reported here are averages over 20 repeated runs, and the projection dimension $d = 10$. Still the proposed SEPCA presents the best performance among the compared approaches. But different from the synthetic experiments, LDA does not work well on these real data sets.

The results on both synthetic and real data show that SEPCA outperforms the other four approaches. This might be attributed to its adaptive exponential family model approximation and its global optimization, while SDR_GLM and SPPCA apparently suffer from local optima.

## 6 Conclusions

In this paper, we propose a supervised exponential family PCA (SEPCA) approach, which can be solved efficiently to find global solutions. Moreover, SEPCA overcomes the limitation of the Gaussian assumption of PCA and SPPCA by using a data adaptive approximation for exponential family models. A simple, straightforward projection method for new testing data has also been constructed. Empirical study suggests that this SEPCA outperforms other supervised dimensionality reduction approaches, such as SDR_GLM, SPPCA, LDA and colored MVU.

Table 1: Data set statistics and test accuracy results (%)

| Dataset | #Data | #Dim | #Class | FULL | SEPCA | SDR_ GLM | SPPCA | LDA | colored MVU |
|---|---|---|---|---|---|---|---|---|---|
| Yale | 165 | 4096 | 15 | 65.3 | 64.4 | 58.8 | 51.6 | 31.0 | 21.1 |
| YaleB | 2414 | 1024 | 38 | 47.0 | 20.5 | 19.0 | 9.8 | 6.2 | 2.8 |
| 11 Tumor | 174 | 12533 | 11 | 77.6 | 88.9 | 63.5 | 63.0 | 23.7 | 40.2 |
| Usps3456 | 120 | 256 | 4 | 82.1 | 79.7 | 77.9 | 78.5 | 74.3 | 75.8 |
| Newsgroup | 19928 | 25284 | 20 | 32.1 | 16.9 | – | 6.9 | 10.0 | 10.4 |

# References

[1] A. Belloni. Introduction to bundle methods. Technical report, MIT, 2005.

[2] J. Borwein and A. Lewis. *Convex Analysis and Nonlinear Optimization.* Springer, 2000.

[3] S. Boyd and L. Vandenberghe. *Convex Optimization.* Cambridge U. Press, 2004.

[4] M. Collins, S. Dasgupta, and R. Schapire. A generalization of principal component analysis to the exponential family. In *Advances in Neural Information Processing Systems (NIPS)*, 2001.

[5] R. Fisher. The use of multiple measurements in taxonomic problems. *Annals of Eugenics*, 7:179–188, 1936.

[6] Y. Guo and D. Schuurmans. Convex relaxations of latent variable training. In *Advances in Neural Information Processing Systems (NIPS)*, 2007.

[7] Y. Guo and D. Schuurmans. Efficient global optimization for exponential family PCA and low-rank matrix factorization. In *Allerton Conf. on Commun., Control, and Computing*, 2008.

[8] I. Jolliffe. *Principal Component Analysis.* Springer Verlag, 2002.

[9] N. Lawrence. Probabilistic non-linear principle component analysis with gaussian process latent variable models. *Journal of Machine Learning Research*, 6:1783–1816, 2005.

[10] S. Mika, G. Ratsch, J. Weston, B. Scholkopf, and K. Muller. Fisher discriminant analysis with kernels. In *IEEE Neural Networks for Signal Processing Workshop*, 1999.

[11] M. Overton and R. Womersley. Optimality conditions and duality theory for minimizing sums of the largest eigenvalues of symmetric matrices. *Math. Prog.*, 62:321–357, 1993.

[12] I. Rish, G. Grabarnilk, G. Cecchi, F. Pereira, and G. Gordon. Closed-form supervised dimensionality reduction with generalized linear models. In *Proceedings of International Conference on Machine Learning (ICML)*, 2008.

[13] Sajama and A. Orlitsky. Semi-parametric exponential family PCA. In *Advances in Neural Information Processing Systems (NIPS)*, 2004.

[14] Sajama and A. Orlitsky. Supervised dimensionality reduction using mixture models. In *Proceedings of the International Conference on Machine Learning (ICML)*, 2005.

[15] L. Song, A. Smola, K. Borgwardt, and A. Gretton. Colored maximum variance unfolding. In *Advances in Neural Information Processing Systems (NIPS)*, 2007.

[16] M. Tipping and C. Bishop. Probabilistic principal component analysis. *Journal of the Royal Statistical Society, B*, 6(3):611–622, 1999.

[17] P. Tseng. Convergence of a block coordinate descent method for nondifferentiable minimization. *Journal of Optimization Theory and Applications*, 109:457–494, 2001.

[18] M. Wainwright and M. Jordan. Graphical models, exponential families, and variational inference. Technical Report TR-649, UC Berkeley, Dept. Statistics, 2003.

[19] S. Yu, K. Yu, V. Tresp, H. Kriegel, and M. Wu. Supervised probabilistic principal component analysis. In *Proceedings of 12th ACM SIGKDD International Conf. on KDD*, 2006.
